# Information Factorization in Connectionist Models of Perception

**Javier R. Movellan**
Department of Cognitive Science
Institute for Neural Computation
University of California San Diego

**James L. McClelland**
Center for the Neural Bases of Cognition
Department of Psychology
Carnegie Mellon University

## Abstract

We examine a psychophysical law that describes the influence of stimulus and context on perception. According to this law choice probability ratios factorize into components independently controlled by stimulus and context. It has been argued that this pattern of results is incompatible with feedback models of perception. In this paper we examine this claim using neural network models defined via stochastic differential equations. We show that the law is related to a condition named channel separability and has little to do with the existence of feedback connections. In essence, channels are separable if they converge into the response units without direct lateral connections to other channels and if their sensors are not directly contaminated by external inputs to the other channels. Implications of the analysis for cognitive and computational neurosicence are discussed.

## 1 Introduction

We examine a psychophysical law, named the Morton-Massaro law, and its implications to connectionist models of perception and neural information processing. For an example of the type of experiments covered by the Morton-Massaro law consider an experiment by Massaro and Cohen (1983) in which subjects had to identify synthetic consonant sounds presented in the context of other phonemes. There were two response alternatives, seven stimulus conditions, and four context conditions. The response alternatives were /l/ and /r/, the stimuli were synthetic sounds generated by varying the onset frequency of the third formant, followed by the vowel /i/. Each of the 7 stimuli was placed after each of four different context consonants, /v/, /s/, /p/, and /t/. Morton (1969) and Massaro independently showed that in a remarkable range of experiments of this type, the influence of stimulus and context on response probabilities can be accounted for with a factorized version of Luce's strength model (Luce, 1959)

$$P(R = k \mid S = i, C = j) \quad = \quad \frac{\eta_S(i,k)\,\eta_C(j,k)}{\sum_l \eta_S(i,l)\,\eta_C(j,l)}, \text{ for } (i,j,k) \in \mathcal{S} \times \mathcal{C} \times \mathcal{R}. \quad (1)$$

Here $S$, $C$ and $R$ are random variables representing the stimulus, context and the subject's response, $\mathcal{S}$, $\mathcal{C}$ and $\mathcal{R}$ are the set of stimulus, context and response al-

ternatives, $\eta_S(i,k) > 0$ represents the support of stimulus $i$ for response $k$, and $\eta_C(j,k) > 0$ the support of context $j$ for response $k$. Assuming no strength parameter is exactly zero, (1) is equivalent to

$$\frac{P(R=k \mid S=i, C=j)}{P(R=l \mid S=i, C=j)} = \left(\frac{\eta_S(i,k)}{\eta_S(i,l)}\right)\left(\frac{\eta_C(j,k)}{\eta_C(j,l)}\right), \text{ for all } (i,j,k) \in \mathcal{S} \times \mathcal{C} \times \mathcal{R}.$$

(2)

This says that response probability ratios factorize into two components, one which is affected by the stimulus but unaffected by the context and one affected by the context but unaffected by the stimulus.

## 2  Diffusion Models of Perception

Massaro (1989) conjectured that the Morton-Massaro law may be incompatible with feedback models of perception. This conjecture was based on the idea that in networks with feedback connections the stimulus can have an effect on the context units and the context can have an effect on the stimulus units making it impossible to factorize the influence of information sources. In this paper we analyze such a conjecture and show that, surprisingly, the Morton-Massaro law has little to do with the existence of feedback and lateral connections. We ground our analysis on continuous stochastic versions of recurrent neural networks [1]. We call these models diffusion (neural) networks for they are stochastic diffusion processes defined by adding Brownian motion to the standard recurrent neural network dynamics. Diffusion networks are defined by the following stochastic differential equation

$$dY_i(t) = \mu_i(Y(t), X)\, dt + \sigma\, dB_i(t) \quad \text{for } i \in \{1, \cdots, n\},$$

(3)

where $Y_i(t)$ is a random variable representing the *internal potential* at time $t$ of the $i^{th}$ unit, $Y(t) = (Y_1(t), \cdots, Y_n(t))'$, $X$ represents the external input, which consists of stimulus and context, and $B_i$ is Brownian motion, which acts as a stochastic driving term. The constant $\sigma > 0$, known as the *dispersion*, controls the amount of noise injected onto each unit. The function $\mu_i$, known as the *drift*, determines the average instantaneous change of activation and is borrowed from the standard recurrent neural network literature: this change is modulated by a matrix $w$ of connections between units, and a matrix $v$ that controls the influence of the external inputs onto each unit.

$$\mu_i(Y_i(t), X) = \frac{1}{\kappa_i(Y_i(t))}(\bar{Y}_i(t) - Y_i(t)), \quad \text{for all } i \in \{1, \cdots, n\},$$

(4)

where $1/\kappa_i$ is a positive function, named the capacitance, controlling the speed of processing and

$$\bar{Y}_i(t) = \sum_j w_{i,j}\, Z_j(t) + \sum_k v_{i,k} X_k, \quad \text{for all } i \in \{1, \cdots, n\},$$

(5)

$$Z_j(t) = \varphi_i(Y_j(t)) = \varphi(\alpha_i Y_j(t)) = 1/(1 + e^{-\alpha_i Y_i(t)}).$$

(6)

Here $w_{i,j}$, an element of the connection matrix $w$, is the weight from unit $j$ to unit $i$, $v_{i,k}$ is an element of the matrix $v$, $\varphi$ is the logistic activation function and the $\alpha_i > 0$ terms are *gain* parameters, that control the sharpness of the activation functions. For large values of $\alpha_i$ the activation function of unit $i$ converges to a step function. The variable $Z_j(t)$ represents a short-time mean firing rate (the activation) of unit

$j$ scaled in the $(0, 1)$ range. Intuition for equation (4) can be achieved by thinking of it as a the limit of a discrete time difference equation, in such case

$$Y(t + \Delta t) = Y_i(t) + \mu_i(Y_i(t), X)\Delta t + \sigma\sqrt{\Delta t}N_i(t), \tag{7}$$

where the $N_i(t)$ are independent standard Gaussian random variables. For a fixed state at time $t$ there are two forces controlling the change in activation: the drift, which is deterministic, and the dispersion which is stochastic. This results in a distribution of states at time $t + \Delta t$. As $\Delta t$ goes to zero, the solution to the difference equation (7) converges to the diffusion process defined in (4). In this paper we focus on the behavior of diffusion networks at stochastic equilibrium, i.e., we assume the network is given enough time to approximate stochastic equilibrium before its response is sampled.

## 3 Channel Separability

In this section we show that the Morton-Massaro is related to an architectural constraint named channel separability, which has nothing to do with the existence of feedback connections. In order to define channel separability it is useful to characterize the function of different units using the following categories: 1) *Response specification units*: A unit is a response specification unit, if, when the state of all the other units in the network is fixed, changing the state of this unit affects the probability distribution of overt responses. 2) *Stimulus units*: A unit belongs to the stimulus channel if : a) it is not a response unit, and b) when the state of the response units is fixed, the probability distribution of the activations of this unit is affected by the stimulus. 3) *Context units*: A unit belongs to the context channel if: a) it is not a response unit, and b) when the states of the response units are fixed, the probability distribution of the activations of this unit can be affected by the context. Given the above definitions, we say that a network has *separable stimulus and context channels* if the stimulus and context units are disjoint: no unit simultaneously belongs to the stimulus and context channels. In essence, channels are structurally separable if they converge into the response units without direct lateral connections to other channels and if their sensors are not directly contaminated by external inputs to the other channels (see Figure 1).

In the rest of the paper we show that if a diffusion network is structurally separable the Morton-Massaro law can be approximated with arbitrary precision regardless of the existence of feedback connections. For simplicity we examine the case in which the weight matrix is symmetric. In such case, each state has an associated goodness function that greatly simplifies the analysis. In a later section we discuss how the results generalize to the non-symmetric case.

Let $y \in \mathbb{R}^n$ represent the internal potential of a diffusion network. Let $z_i = \varphi(\alpha_i y_i)$ for $i = 1, \cdots, n$ represent the firing rates corresponding to $y$. Let $z^s$, $z^c$ and $z^r$ represent the components of $z$ for the units in the stimulus channel, context channel and response specification module. Let $x$ be a vector representing an input and let $x^s$, $x^c$ be the components of $x$ for the external stimulus and context. Let $\alpha = (\alpha_1, \cdots, \alpha_n)$ be a fixed gain vector and $Z^\alpha(t)$ a random vector representing the firing rates at time $t$ of a network with gain vector $\alpha$. Let $Z^\alpha = \lim_{t \to \infty} Z^\alpha(t)$, represent the firing rates at stochastic equilibrium. In Movellan (1998) it is shown that if the weights are symmetric i.e., $w = w'$ and $1/\kappa_i(x) = d\varphi_i(x)/dx$ then the equilibrium probability density of $Z^\alpha$ is as follows

$$p_{Z^\alpha|X}(z^s, z^c, z^r \mid x^s, x^c) = \frac{1}{K_\alpha(x_s, x_c)}\exp((2/\sigma^2)\,G_\alpha(z^s, z^r \mid x_s, x_c)), \tag{8}$$

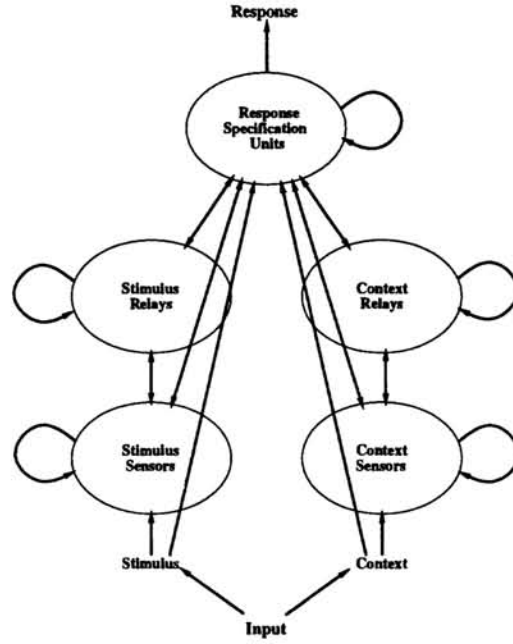

Figure 1: A network with separable context and stimulus processing channels. The stimulus sensor and stimulus relay units make up the stimulus channel units, and the context sensor and context channel units make up the context channel units. Note that any of the modules can be empty except the response module.

where

$$K_\alpha(x_s, x_c) = \int \exp((2/\sigma^2)\, G_\alpha(z \mid x_s, x_c))\, dz, \tag{9}$$

$$G_\alpha(z \mid x) = H(z \mid x) - \sum_{i=1}^{n} S_{\alpha_i}(z_i), \tag{10}$$

$$H(z \mid x) = z'\, w\, z/2 + z'\, v\, x, \tag{11}$$

$$S_{\alpha_i}(z_i) = \alpha_i \left( \log(z_i) + \log(1 - z_i) \right) + \frac{1}{\alpha_i} \left( z_i \log(z_i) + (1 - z_i) \log(1 - z_i) \right). \tag{12}$$

Without loss of generality hereafter we set $\sigma^2 = 2$. When there are no direct connections between the stimulus and context units there are no terms in the goodness function in which $x^s$ or $z^s$ occur jointly with $x^c$ or $z^c$. Because of this, the goodness can be separated into three additive terms, that depend on $x^s$, $x^c$ and a third term which depends on the response units:

$$G_\alpha(z^s, z^c, z^r \mid x^s, x^c) = G_\alpha^s(z^s, z^r \mid x^s) + G_\alpha^c(z^r, z^c \mid x^c) + G_\alpha^r(z^r), \tag{13}$$

where

$$G_\alpha^s(z^s, z^r \mid x^s) = (z^s)' w_{s,s} z^s/2 + (z^s)' w_{s,r} z^r + (z^s)' v_{s,s} x^s + (z^r)' v_{r,s} x^s - \sum_i S(z_i^s), \tag{14}$$

$$G_\alpha^c(z^c, z^r \mid x^s) = (z^c)' w_{c,c} z^c/2 + (z^c)' w_{c,r} z^r + (z^c)' v_{c,c} x^c + (z^r)' v_{r,c} x^c - \sum_i S(z_i^c), \tag{15}$$

$$G_\alpha^r(z^r) = (z_i^r)' w_{r,r} z^r/2 - \sum_i S(z_i^r). \tag{16}$$

where $w_{s,r}$ is a submatrix of $w$ connecting the stimulus and response units. Similar notation is used for the other submatrices of $w$ and $v$. It follows that we can write the ratio of the joint probability density' of two states $z$ and $\tilde{z}$ as follows:

$$\frac{p_{Z_\alpha|X}(z^s, z^c, z^r \mid x^s, x^c)}{p_{Z_\alpha|X}(\tilde{z}^s, \tilde{z}^c, \tilde{z}^r \mid x^s, x^c)} = \frac{\exp(G_\alpha^s(z^s, z^r \mid x_s) + G_\alpha^c(z^c, z^r \mid x^c) + G_\alpha^r(z^r))}{\exp(G_\alpha^s(\tilde{z}^s, \tilde{z}^r \mid x_s) + G_\alpha^c(\tilde{z}^c, \tilde{z}^r \mid x^c) + G_\alpha^r(\tilde{z}^r))} , \quad (17)$$

which factorizes as desired. To get probability densities for the response units, we integrate over the states of all the other units

$$p_{Z_\alpha^r|X}(z^r \mid x^s, x^c) = \int \int p_{Z_\alpha|X}(z^s, z^c, z^r \mid x^s, x^c) \, dz^s \, dz^c, \quad (18)$$

and after rearranging terms

$$p_{Z_\alpha^r|X}(z^r \mid x^s, x^c) = \frac{1}{K_\alpha(x^s, x^c)} \left( \int \exp(G_x(z^s, z^r \mid x^s) + G_r(z^r)) \, dz^s \right) \left( \int \exp(G_c(z^c, z^r \mid x^c)) \, dz^c \right), \quad (19)$$

which also factorizes. All is left is mapping continuous states of the response units to discrete external responses. To do so we partition the space of the response specification units into discrete regions. The probability of a response becomes the integral of the probability density over the region corresponding to that response. The problem is that the integral of probability densities does not necessarily factorize even though the densities factorize at every point.

Fortunately there are two important cases for which the law holds, at least as a good approximation. The first case is when the response regions are small and thus we can approximate the integral over that region by the density at a point times the volume of the region. In such a case the ratio of the integrals can be approximated by the ratio of the probability densities of those individual states. The second case applies to models, like McClelland and Rumelhart's (1981) interactive activation model, in which each response is associated with a distinct response unit. These models typically have negative connections amongst the response units so that at equilibrium one unit tends to be active while the others are inactive. In such a case a common response policy picks the response corresponding to the active unit. We now show that such a policy can approximate the Morton-Massaro law to an arbitrary level of precision as the gain parameter of the response units is increased. Let $z$ represent the joint state of a network and let the first $r$ components of $z$ be the states of the response specification units. Let $z^{(1)} = (1, 0, 0, \cdots, 0)'$, $z^{(2)} = (0, 1, 0, \cdots, 0)'$ be two $r$-dimensional vectors representing states of the response specification units. For $i \in \{1, 2\}$ and $\Delta \in (0, 1)$ let

$$z_\Delta^{(i)} = (1 - z^{(i)})\Delta + (z^{(i)})(1 - \Delta), \quad (20)$$

$$R_\Delta^{(i)} = \{x \in \mathbb{R}^r : x_j \in ((1 - \Delta)z_j^{(i)}, \Delta + (1 - \Delta)z_j^{(i)}), \text{ for } j = 1, \cdots, r\}. \quad (21)$$

The sets $R_\Delta^{(1)}$ and $R_\Delta^{(2)}$ are regions of the $[0, 1]^r$ space mapping into two distinct external responses. We now investigate the convergence of the probability ratio of these two responses as we let $\Delta \to 0$, i.e., as the response regions collapse into corners of $[0, 1]^r$.

$$\lim_{\Delta \to 0} \frac{P(Z_\alpha^r \in R_\Delta^{(2)} \mid X = x)}{P(Z_\alpha^r \in R_\Delta^{(1)} \mid X = x)} = \lim_{\Delta \to 0} \frac{\int_{R_\Delta^{(2)}} p_{Z_\alpha^r|X}(u \mid x) du}{\int_{R_\Delta^{(1)}} p_{Z_\alpha^r|X}(u \mid x) du} = \quad (22)$$

$$\lim_{\Delta \to 0} \frac{\Delta^r p_{Z_\alpha^r|X}(z_\Delta^{(2)} \mid x)}{\Delta^r p_{Z_\alpha^r|X}(z_\Delta^{(1)} \mid x)} = \lim_{\Delta \to 0} \frac{\int \int e^{G_\alpha^r(z_\Delta^{(2)}, z^s, z^c \mid x)} dz^s \, dz^c}{\int \int e^{G_\alpha(z_\Delta^{(1)}, z^s, z^c \mid x)} dz^s \, dz^c}. \quad (23)$$

Table 1: Predictions by the Morton-Massaro law (left side) versus diffusion network (square brackets) for subject 7 of Massaro and Cohen (1983) Experiment 2. Each prediction of the diffusion network is based on 100 random samples.

| Stimulus | Context | | | | | | | |
|---|---|---|---|---|---|---|---|---|
| | V | | S | | P | | T | |
| 0 | 0.0017 | [0.01] | 0.0000 | [0.00] | 0.0152 | [0.03] | 0.9000 | [0.91] |
| 1 | 0.0126 | [0.00] | 0.0000 | [0.00] | 0.1008 | [0.10] | 0.9849 | [0.97] |
| 2 | 0.1105 | [0.19] | 0.0008 | [0.00] | 0.5208 | [0.45] | 0.9984 | [1.00] |
| 3 | 0.5463 | [0.54] | 0.0079 | [0.00] | 0.9133 | [0.91] | 0.9998 | [1.00] |
| 4 | 0.9827 | [1.00] | 0.2756 | [0.30] | 0.9980 | [1.00] | 0.9999 | [1.00] |
| 5 | 0.9999 | [1.00] | 0.9924 | [0.99] | 0.9999 | [1.00] | 1.0000 | [1.00] |
| 6 | 0.9999 | [1.00] | 0.9924 | [1.00] | 0.9999 | [1.00] | 1.0000 | [1.00] |

Now note that

$$G_\alpha(z_\Delta^{(1)}, z^s, z^c \mid x) = H(z_\Delta^{(1)}, z^s, z^c \mid x) - \sum_{i=1}^{r} S_{\alpha_i}(z_{\Delta,i}^{(1)}) - \sum_i S_{\alpha_i}(z_i^s) - \sum_j S_{\alpha_j}(z_j^c),$$

(24)

and since $\sum_{i=1}^{r} S_{\alpha_i}(z_{\Delta,i}^{(1)}) = \sum_{i=1}^{r} S_{\alpha_i}(z_{\Delta,i}^{(2)})$, it follows that

$$\lim_{\Delta \to 0} \frac{P(Z_\alpha^r \in R_\Delta^{(2)} \mid X = x)}{P(Z_\alpha^r \in R_\Delta^{(1)} \mid X = x)} = \frac{\int \int e^{H(z_\Delta^{(2)}, z^s, z^c \mid x) - \sum_i S_{\alpha_i}(z_i^s) - \sum_j S_{\alpha_j}(z_j^c)} dz^s \, dz^c}{\int \int e^{H(z_\Delta^{(1)}, z^s, z^c \mid x) - \sum_i S_{\alpha_i}(z_i^s) - \sum_j S_{\alpha_j}(z_j^c)} dz^s \, dz^c}.$$

(25)

It is easy to show that this ratio factorizes. Moreover, for all $\Delta > 0$ if we let $\alpha_1 = \cdots = \alpha_r = \alpha$, where $\alpha > 0$ then

$$\lim_{\alpha \to \infty} P(Z_\alpha^r \in [\Delta, 1 - \Delta]^r) = 0,$$

(26)

since as the gain of the response units increases $S_{\alpha_i}$ decreases very fast at the corners of $(0,1)^r$. Thus as $\alpha \to \infty$ the random variable $Z_\alpha^r$ converges in distribution to a discrete random variable with mass at the corner of the $[0,1]^r$ hypercube and with factorized probability ratios as expressed on (25). Since the indexing of the response units is arbitrary the argument applies to all the responses.

□

## 4   Discussion

Our analysis establishes that in diffusion networks the Morton-Massaro law is not incompatible with the presence of feedback and lateral connections. Surprisingly, even though in diffusion networks with feedback connections stimulus and context units are interdependent, it is still possible to factorize the effect of stimulus and context on response probabilities.

The analysis shows that the Morton-Massaro can be arbitrarily approximated as the sharpness of the response units is increased. In practice we have found very good approximations with relatively small values of the sharpness parameter (see Table 1 for an example). The analysis assumed that the weights were symmetric. Mathematical analysis of the general case with non-symmetric weights is difficult.

However useful approximations exist (Movellan & McClelland, 1995) showing that if the noise parameter $\sigma$ is relatively small or if the activation function $\varphi$ is approximately linear, symmetric weights are not needed to exhibit the Morton-Massaro law.

The analysis presented here has potential applications to investigate models of perception and the functional architecture of the brain. For example the interactive activation model of word perception has a separable architecture and thus, diffusion versions of it adhere to the Morton Massaro law. The analysis also points to potential applications in computational neuroscience. It would be of interest to study whether the Morton-Massaro holds at the level of neural responses. For example, we may excite a neuron with two different sources of information and observe its short term average response to combination of stimuli. If the observed distribution of responses exhibits the Morton-Massaro law, this would be consistent with the existence of separable channels converging into that neuron. Otherwise, it would indicate that the channels from the two input areas to the response may not be structurally separable.

## Footnotes

[1]For an analysis grounded on discrete time networks with binary states see McClelland (1991).

# References

Luce, R. D. (1959). *Individual choice behavior*. New York: Wiley.

Massaro, D. W. (1989). Testing between the TRACE Model and the fuzzy logical model of speech perception. *Cognitive Psychology, 21*, 398–421.

Massaro, D. W. (1998). *Perceiving Talking Faces*. Cambridge, Massachusetts: MIT Press.

Massaro, D. W. & Cohen, M. M. (1983a). Phonological constraints in speech perception. *Perception and Psychophysics, 34*, 338–348.

McClelland, J. L. (1991). Stochastic interactive activation and the effect of context on perception. *Cognitive Psychology, 23*, 1–44.

Morton, J. (1969). The interaction of information in word recognition. *Psychological Review, 76*, 165–178.

Movellan, J. R. (1998). A Learning Theorem for Networks at Detailed Stochastic Equilibrium. *Neural Computation, 10*(5), 1157–1178.

Movellan, J. R. & McClelland, J. L. (1995). Stochastic interactive processing, channel separability and optimal perceptual inference: an examination of Morton's law. Technical Report PDP.CNS.95.4, Available at http://cnbc.cmu.edu, Carnegie Mellon University.